# QVAE-Mole: The Quantum VAE with Spherical Latent Variable Learning for 3-D Molecule Generation

**Huanjin Wu**[†], **Xinyu Ye**[†], **Junchi Yan**[*]
Dept. of CSE & School of AI & MOE Key Lab of AI, Shanghai Jiao Tong University
{whj1201,xinyu_ye,yanjunchi}@sjtu.edu.cn

## Abstract

Molecule generation ideally in its 3-D form has enjoyed wide applications in material, chemistry, life science, etc. We propose the first quantum parametric circuit for 3-D molecule generation for its potential quantum advantage especially considering the arrival of Noisy Intermediate-Scale Quantum (NISQ) era. We choose the Variational AutoEncoder (VAE) scheme for its simplicity and one-shot generation ability, which we believe is more quantum-friendly compared with the auto-regressive generative models or diffusion models as used in classic approaches. Specifically, we present a quantum encoding scheme designed for 3-D molecules with qubits complexity $\mathcal{O}(C \log n)$ ($n$ is the number of atoms) and adopt a von Mises-Fisher (vMF) distributed latent space to meet the inherent coherence of the quantum system. We further design to encode conditions into quantum circuits for property-specified generation. Experimentally, our model could generate plausible 3-D molecules and achieve competitive quantitative performance with significantly reduced circuit parameters compared with their classic counterparts.

## 1 Introduction

Beyond molecule graph generation, 3-D molecule generation which can often be more challenging yet of practical value, e.g. for drug design, has received wide attention in recent years [1]. On the one hand, the space of possible molecules and chemical compounds is vast, often described as a "chemical space" with an immense number of dimensions, and data-driven methods relying on machine learning (ML) have been introduced [2, 3]. On the other hand, quantum computing has demonstrated strong expressive capabilities in various learning and optimization applications, including solving [4], classification [5], and discovery [6]. Particularly, it has potential advantages in tasks related to the microphysical world, such as chemistry simulation [7], prediction of molecular properties [8], and approximation of ground-state energy [9]. Therefore, beyond classic ML, here we dive into the quantum world for 3-D molecule generation with the arrival of the so-called NISQ era. However, limited by the current development of quantum hardware, quantum machine learning (QML) models, particularly quantum generative models, are still in their infancy stages, especially when compared to the well-developed classic neural models, thus the current performance of QML models may not match that of SOTA classic counterparts [10–13].

Recent efforts have been made to introduce quantum methods for molecule design. QGAN-HG [14] is a hybrid model based on Generative Adversarial Network (GAN), consisting of a classic discriminator and a hybrid generator. However, hybrid models cannot be implemented on NISQ devices, and

---

[*]Corresponding author. [†] Equal contribution. This work was partly supported by NSFC (92370201, 62222607) and Shanghai Municipal Science and Technology Major Project under Grant 2021SHZDZX0102.

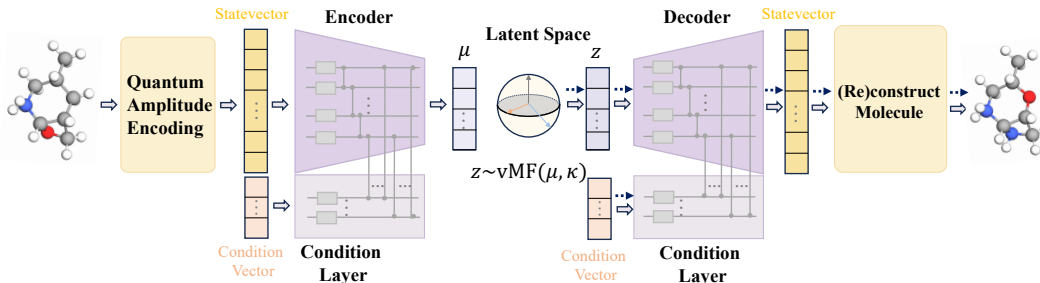

Figure 1: **Pipeline of QVAE-Mole and QCVAE-Mole with a vMF distributed latent space.** We first use amplitude encoding to get the initial quantum state vectors from classic data of molecules. Then the quantum encoder learns the mean direction $\mu$ in the latent space, which is used to sample a latent variable $z_i \sim \text{vMF}(\mu, \kappa)$. A subsequent quantum decoder decodes $z$ trying to match the input state vector. Then molecule is reconstructed from the output state vector of decoder. For conditional generation, we have a condition vector for both input data and latent space, then we use condition qubits as well as condition layers to encode the given conditions into the quantum circuit. Solid and dashed arrows represent the training and inference phase, respectively.

QGAN-HG only utilizes quantum circuits to learn features with low dimensions, which actually decreases the performance of classic neural networks. Another work SQ-VAE [15], a quantum Variational AutoEncoder (VAE) approach, represents the molecular graph as a flattened adjacency matrix and converts it into a quantum state through amplitude encoding. However, it is difficult for its quantum circuit to reconstruct the topology structure by the inputted flattened adjacency matrices [16]. Furthermore, SQ-VAE adopts a normal distributed latent space, where latent variables remain unconstrained by normalization requirements. Nevertheless, the quantum state outputs of SQ-VAE are subject to normalization, necessitating a linear mapping with classic parameters to map the output into non-normalized latent variables. This indicates the incompatibility of the normal distribution with quantum systems. In all, these two quantum methods are limited to generating molecular graphs and fall short of generating 3-D molecules.

To address the above issues and explore a quantum version of VAE for *3-D data generation* (which is the first time in literature), we develop a full quantum VAE framework, especially for 3-D molecule generation. Firstly, we introduce a quantum encoding scheme designed for 3-D molecules to fulfill the normalization constraints by quantum states. This involves normalizing the 3-D coordinates, atom types, and an auxiliary value associated with each atom. These normalized atom vectors are then concatenated to form the initial quantum state. Secondly, we adopt von Mises-Fisher (vMF) distribution, which lies in a hyperspherical space and can inherently meet the restrictions of quantum systems (the norm of quantum state is 1).

In addition, the ultimate objective is to generate molecules directly with desired properties in various domains [17]. For instance, there is a growing demand for molecules with a low HOMO-LUMO gap in the field of organic semiconductor development [18]. To this end, we further present a *conditional version* of quantum VAE named QCVAE-Mole, which have the ability to generate molecules with multiple desired properties. Specifically, we introduce condition qubits as well as condition parametric layers to encode given conditions into the proposed quantum ansatz, thus we can generate molecules with desired properties by giving specific condition vectors. **The contributions are:**

1) We propose the first fully (to our best knowledge) quantum VAE for *3-D data generation* and detailed quantum circuits compatible with NISQ devices. For generated quantum states, we fulfill its inherent normalization requirement via the vMF distribution in a spherical latent space.

2) Our conditional VAE version manages to encode the conditions into the quantum circuit, which is, to our best knowledge, the first fully quantum circuit capable of *conditional* VAE-based generation.

3) We conduct all the experiments in a TorchQuantum-based simulation environment in line with many QML works [8, 19, 20]. Extensive results on the QM9 benchmark show that our model outperforms all other quantum (or hybrid) methods [14, 15] and delivers comparable results to several classic methods [21–23] with significantly reduced parameters.

## 2 Preliminaries

**Quantum machine learning.** A quantum bit (qubit) is the unit of quantum information and computing, which exists in the superposition state of both 0 and 1, as denoted by $|\psi\rangle = \alpha|0\rangle + \beta|1\rangle$, where $\alpha$ and $\beta$ are complex coefficients. $|\alpha|^2$ and $|\beta|^2$ are the probability amplitudes for the qubit being in basis states ($|0\rangle$ or $|1\rangle$). For $n$-qubit system, there are $2^n$ basis states. Quantum gates are the building blocks that manipulate quantum states. These gates are represented by unitary matrices. Parameterized quantum gates are quantum gates with one or more parameters. See Appendix B for the single-qubit and double-qubit parameterized quantum gates used in this paper.

The concept of Variational Quantum Algorithms (VQA) was proposed by [24], which utilizes quantum advantages to solve ML problems on NISQ devices. Then, Parameterized Quantum Circuits (PQCs) serve as the specific implementations of these VQAs, with the parameterized quantum gates mentioned above being key components of PQCs. The parameters $\theta$ can be optimized by a classic optimizer to minimize loss function $\mathcal{L}(\theta)$, which evaluates the dissimilarity between the output of QPC and the target result. Even if the measurement itself does not provide gradient information, the gradients of $\theta$ can be directly estimated by perturbing $\theta$ slightly. For instance, the Parameter Shift Rule [25] is a popular technique used to achieve the gradients in many QML models [19, 26]. Also using gradient backpropagation, classic learning models are adapted into their quantum version, e.g. QCNN [27], QRNN [19], QGAN [12], QLSTM [28], and etc, They are highly recognized for their intellectual novelty as well as their potential in the NISQ era.

**Variational autoencoder.** Variational autoencoders (VAEs) [21] are a class of generative models that combine autoencoders with variational inference techniques. VAEs provide a principled approach to learning latent representations and generating new data samples. Formally, VAE is represented by an inference network (*i.e.*, encoder) $q_\phi(z|x)$ and a generator network (*i.e.*, decoder) $p_\theta(x|z)$, where $z$ denotes the latent variables. The intractable true posterior $p_\theta(z|x)$ is approximated by the inference network $q_\phi(z|x)$, which is parameterized by a neural network and outputs a probability distribution for each data point $x$. Given the training set $D = \{x_i\}_{i=1}^N$ and the prior $p(\mathbf{z})$, the final objective is to minimize the negative of Evidence Lower Bound (ELBO):

$$\min_{\theta,\phi} \mathcal{L}_{\text{ELBO}}(\theta, \phi; x) = -\mathbb{E}_{z \sim q_\phi(z|x)}\left[\log\left(p_\theta(x|z)\right)\right] + D_{KL}\left[q_\phi(z|x)\|p(z)\right]. \tag{1}$$

It serves as a proxy for the log-likelihood of the data with a regularizer. The first term is known as reconstruction loss, and the second is the Kullback-Leibler (KL) divergence.

## 3 Methodology

We leave a detailed discussion on related works in Appendix A, including classic methods for molecule generation and quantum generative learning. In this section, we first introduce how to encode classic data into a quantum state and then elaborate on the quantum architecture of QVAE-Mole and QCVAE-Mole. Fig. 1 shows the pipeline.

### 3.1 Encoding 3-D Molecule to Initial Quantum State

We use an attributed point cloud $\{(v_i, a_i)\}_{i=0}^{n-1}$ to represent a 3-D molecule with $n$ atoms, where each point represents an atom. Here $v_i \in \mathbb{R}^3$ denotes the 3-D coordinates of the atom and $a_i \in \{0,1\}^k$ denotes the atom type ($k$ is the number of atom types). To encode the molecules into quantum states, we normalize them in two aspects: 1) Each 3-D molecule can undergo arbitrary translations and rotations, and there is no inherent ordering among atoms, which requires us to fix the order of atoms and the molecular conformation to obtain unique encoding. 2) The amplitudes of quantum states require us to keep the input data of the molecule within the positive octant. See details of normalization in Appendix C.

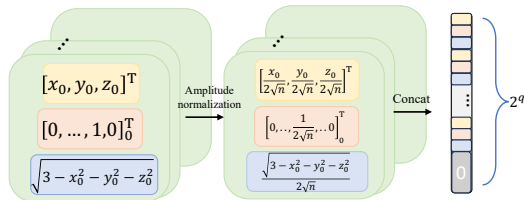

Figure 2: **Classic data encoding.** We encode 3-D molecules with 3-D coordinates and atom types through three steps: 1)introduce an auxiliary value, 2) normalize to a norm 1.0, and 3) convert into a quantum state vector via amplitude encoding.

We will introduce how to encode the normalized 3-D molecules into quantum states. It converts the given classic data sample $x$ into its corresponding quantum state $|\psi_x\rangle$. Here, we choose amplitude encoding, which allows us to utilize the exponentially large Hilbert space compared with angle encoding. The data of length $N$ is encoded into the amplitudes of $q = \lceil \log_2 N \rceil$ qubits. In this case, we need to encode each atom with its 3-D coordinates as well as the one-hot form of atom types.

As amplitude encoding demands a state vector with the unit norm, inspired by [29], here we introduce an auxiliary value $\sqrt{3 - x_i^2 - y_i^2 - z_i^2}$ for each normalized atom $\tilde{v}_i = (x_i, y_i, z_i)$ as a constant normalization factor. Therefore, we need $4 + k$ entries for each atom (3 for 3-D coordinates, $k$ for one hot embedding, 1 for auxiliary value), and the total number of entries is $n * (4 + k)$. See Fig. 2 for our encoding. Furthermore, the state vector size is always a power of 2, so we fill the remaining entries with zeros (padding entries) and obtain a state vector of size $2^q$. The norm of the state vector is 1, thus we need to normalize all values by $2\sqrt{n}$. We get the initial quantum state of a molecule by:

$$
\begin{aligned}
|\psi_0\rangle = \frac{1}{2\sqrt{n}} \sum_{i=0}^{n-1} \Big( & x_i|r_i\rangle + y_i|r_i+1\rangle + z_i|r_i+2\rangle + 1|r_i+3+t_i\rangle \\
& + \sqrt{3 - x_i^2 - y_i^2 - z_i^2}|r_i+k+3\rangle \Big) + \sum_{j=(n*(4+k))}^{2^q-1} 0|j\rangle,
\end{aligned}
\tag{2}
$$

where $r_i = (k+4)*i$ and $t_i$ is the atom type of $i$-th atom. The discussion of the initial state preparation can be found in Appendix C.

**Qubits complexity analysis.** The number of qubits in our proposed framework comes to $q = \lceil \log_2(n_{max} * (4+k)) \rceil = \mathcal{O}(C \log n)$, where $n_{max}$ represents the maximum number of atoms. The qubits complexity analysis of other quantum methods can be seen from Appendix C.

## 3.2 Full Quantum Architecture

We propose QVAE-Mole, a fully quantum circuit-based VAE for 3-D molecule generation. Like many works on QML, e.g. QCNN [27],QGAN [12], QLSTM [28], we follow the architecture of its classic design, the VAE in our case, which generally includes encoder, decoder, and latent space. In fact, proposing a quantum counterpart as well as its detailed quantum circuits compatible with NISQ devices is nontrivial, especially for 3-D data (molecule) generation.

### 3.2.1 Encoder

We present the encoder network of our QVAE-Mole. Similar to classic neural networks, the PQC is built layer by layer, where each layer consists of the same arrangement of quantum gates with different trainable parameters. Fig. 3 depicts the general framework of the quantum encoder ansatz. Denote $\mathbf{U}_s(\boldsymbol{\theta}_s^l), \mathbf{U}_{ent}(\boldsymbol{\theta}_{ent}^l)$ as the $l$-th single-qubit layer and entanglement layer with trainable parameters, respectively. The unitary matrix of the proposed encoder

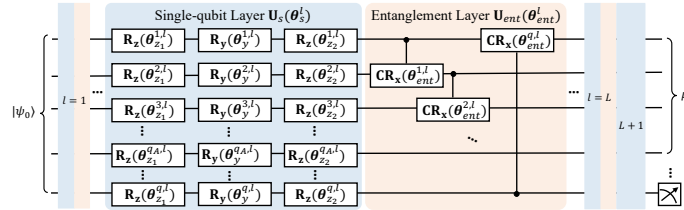

Figure 3: **The ansatz of our quantum encoder**. It takes the initial quantum state $|\psi_0\rangle$ as input and outputs the mean direction $\mu$ of vMF distribution. Each layer includes a single-qubit layer $\mathbf{U}_s$ and an entanglement layer $\mathbf{U}_{ent}$. In the end, we trace out the state of the last $q_B$ qubits: $q_B = q - q_A$.

can be formulated as follows, where $L$ denotes the total number of layers.

$$
\mathbf{U}(\boldsymbol{\theta}) = \mathbf{U}_s(\boldsymbol{\theta}_s^{L+1}) \prod_{l=1}^{L} \Big( \mathbf{U}_{ent}(\boldsymbol{\theta}_{ent}^l) \mathbf{U}_s(\boldsymbol{\theta}_s^l) \Big),
\tag{3}
$$

Specifically, the $l$-the trainable single-qubit layer is formulated as:

$$
\mathbf{U}_s(\boldsymbol{\theta}_s^l) = \bigotimes_{p=1}^{q} \Big( \mathbf{R_z}\big(\boldsymbol{\theta}_{z_1}^{(p,l)}\big) \mathbf{R_y}\big(\boldsymbol{\theta}_y^{(p,l)}\big) \mathbf{R_z}\big(\boldsymbol{\theta}_{z_2}^{(p,l)}\big) \Big),
\tag{4}
$$

where $\boldsymbol{\theta}_{z_1}^{(p,l)}, \boldsymbol{\theta}_{z_2}^{(p,l)}$ denote the parameter of the first and second $\mathbf{R_z}$ gate at the $l$-th layer on the $p$-th qubit, respectively. And the $l$-th entanglement layer can be formulated as:

$$\mathbf{U}_{ent}(\boldsymbol{\theta}_{ent}^l) = \prod_{p=1}^{q} \left( \mathbf{CR_x}\left(p, (p+1)\%q, \boldsymbol{\theta}_{ent}^{(p,l)}\right)\right), \tag{5}$$

where $\mathbf{CR_x}$ is the controlled $\mathbf{R_x}$ gate, $p$ denotes the control qubit, $(p+1)\%q$ denotes the target qubit, and $\boldsymbol{\theta}_{ent}^{(p,l)}$ denotes the corresponding trainable paramenter. The quantum state after the encoder ansatz is $|\psi^E\rangle = \mathbf{U}(\boldsymbol{\theta})|\psi_0\rangle$. Detailed discussions and the theoretical analysis of the expressive power of the designed quantum circuit are shown in Appendix E.

At the end of the encoder circuit, we introduce a measure to reduce the dimension of the output quantum state. Inspired by [30], we divide the encoder circuit into two subsystems, namely subsystem $A$ with $q_A$ qubits and subsystem $B$ with $q_B$ qubits, and $q_A + q_B = q$. Therefore, the output quantum state of the encoder ansatz becomes $|\psi^E\rangle_{AB} = \mathbf{U}(\boldsymbol{\theta})|\psi_0\rangle_{AB}$.

### 3.2.2 Latent space

To convert to latent space, here we discard the information contained in the subsystem $B$ via *tracing out* the state of $q_B$ qubits. This approach integrates quantum non-linearity into the encoder, thereby enriching the transformation process with a layer of complexity beyond that of a mere unitary transformation. Formally, the partial trace is:

$$\rho_A = \text{Tr}_b(\rho_{AB}) = \sum_{j} \left(\mathbf{I}_A \otimes \langle j|_B\right) \rho_{AB} \left(\mathbf{I}_A \otimes |j\rangle_B\right), \tag{6}$$

where $\mathbf{I}_A$ is the identity matrix, $|j\rangle_B \in \left(\mathbb{C}^2\right)^{\otimes q_B}$ are all basis states of subsystem $B$ and $\rho_{AB} = |\psi\rangle_{AB}\langle\psi|_{AB}$. The diagonal of $\rho_A$ contains the squared amplitudes of $|\psi_A\rangle$, as can be converted into the output quantum state of encoder $|\psi^E\rangle_A = \sum_j \sqrt{\rho_{A_{jj}}}|j\rangle$. Then, we perform quantum tomography [31] rather than random measurements on the encoder. The output is the vector of the latent space of dimension $2^{q_A}$. More discussions about tracing out and quantum state tomography are given in Appendix F.

In common VAEs, normal distributions are assumed for the distribution of the latent variables during training. However, normal distributions are unsuitable for data with a latent hyperspherical structure. The quantum state vector requires its L2 norm to be equal to 1, which lies in a hypersphere space. Instead we adopt using a von Mises-Fisher (vMF) distribution, leading to hyperspherical latent space. Formally, the distribution is:

$$q_\phi(z|x) = \text{vMF}(\mu, \kappa) = C_{d,\kappa} e^{\kappa\langle\mu(x),z\rangle}, \tag{7}$$

where $\|\mu\| = 1$ denotes the mean direction. $\kappa$ denotes the concentration parameter, which is commonly set as a constant during training [32]. The normalization constant $C_{d,\kappa}$ is equal to $1/\int_{S^{d-1}} e^{\langle\xi,x\rangle} dS^{d-1}$, where $\xi \in \mathbb{R}^d$ is a predefined parameter vector and $S^{d-1}$ is the sample space $\{x|x \in \mathbb{R}^d, \|x\| = 1\}$. We set that the quantum state $|\psi^E\rangle_A$ seamlessly functions as $\mu$ for learning the vMF distribution. In addition, rejection sampling [33, 32] is utilized to efficiently sample latent variable $z$ from the vMF distribution in the latent space:

$$z \sim \text{vMF}(\||\psi^E\rangle_A|, \kappa) = C_{d,\kappa} e^{\kappa\langle\||\psi^E\rangle_A|, z\rangle}, \tag{8}$$

which represents that the latent variable $z$ is sampled from the vMF distribution with mean direction $\mu = \||\psi^E\rangle_A|$ in the latent space.

### 3.2.3 Decoder

The decoder takes $z$ sampled from the vMF distribution as input and outputs the reconstructed quantum state, which can be further converted to a molecule. The input $z$ has a dimension of $2^{q_A}$, and the reconstructed quantum state should have the same dimension as the initial quantum state, which is $2^q$. This means that the quantum decoder needs to map from a lower-dimensional space to a higher-dimensional one using a unitary transform. We achieve this by first turning $z$ into the state

$|\psi^D\rangle_A$ via amplitude encoding and then expanding $|\psi^D\rangle_A$ with qubits of $B$, which are reset to $|0\rangle_B$. Now we get the initial quantum state of the decoder:

$$|\psi_0^D\rangle = |\psi_0^D\rangle_{AB} = |\psi^D\rangle_A \otimes |0\rangle_B. \tag{9}$$

We design the quantum ansatz of the decoder the same as that of the encoder, then the reconstructed quantum state results in $|\psi_r\rangle = \mathbf{U}(\boldsymbol{\theta}')|\psi_0^D\rangle$, here $\boldsymbol{\theta}'$ denotes the learnable parameters in decoder.

We can get the output quantum state $|\psi_r\rangle$ denoted as $(\alpha_0, \ldots, \alpha_{2^q-1})$. Now, we convert this vector back into classic data, representing a molecule, which serves as the inverse process of encoding:

$$(x', y', z') = (\alpha_{r_i}, \alpha_{r_i+1}, \alpha_{r_i+2}), \tag{10}$$

$$t_i' = \mathrm{argmax}(\alpha_{r_i+3}, \ldots, \alpha_{r_i+3+(k-1)}), \tag{11}$$

$$r_i = (k+4) * i, \quad i = 1, \ldots, n-1, \tag{12}$$

where $x', y', z'$ are the reconstructed coordinates of each atom, $t_i'$ is the reconstructed atom type, and $n$ is the number of atoms. Note that $n$ is known in training but can be arbitrary at inference. Therefore, we need to infer the number of atoms in the generated molecule from the output state vector. We set the following criteria: if $(\alpha_{r+3} + \ldots + \alpha_{r+3+(k-1)}) < T$ for a certain $i$, we consider all subsequent entries to be padding items instead of carrying valid information, and the number of atoms in this generated molecule comes to $n = i$. The hyperparameter $T$ denotes the threshold.

**Remark.** We further discuss why choosing amplitude encoding instead of angle encoding from the perspective of encoding and reconstruction. For angle encoding, we obtain the initial quantum state by converting input information into the rotation angles of qubits. Although it is friendly to initial state preparation, it becomes intractable to reconstruct the input angles from the entangled quantum state, which is the output of the decoder. On the contrary, if we encode the input into the amplitudes of qubits, we can directly reconstruct the input information from the output quantum state vector.

### 3.2.4 Training

Recall the objective function in Eq. 1, we use a uniform vMF prior $p(z) = \mathrm{vMF}(\cdot, 0)$ on the latent space. The vMF prior prevents the KL collapse typically observed in Gaussian VAE settings [34]. In fact, the KL term in our loss term is constant and only depends on the chosen variance $\kappa$, which is a hyperparameter in our model. Thus, we can simplify the Eq. 1 to:

$$\min_{\theta,\phi} \mathcal{L}_{\mathrm{ELBO}}(\theta, \phi; x) = -\mathbb{E}_{z \sim q_\phi(z|x)}[\log(p_\theta(x|z)). \tag{13}$$

In other words, we only need to calculate the reconstruction loss. Here, we can calculate the loss of $|\psi_0\rangle$ and $|\psi_r\rangle$ in two ways. One is to design the reconstruction loss based on converted classic data, while the other is to construct the loss using the fidelity of the quantum state (we denoted as *fidelity loss* in experiments). The former can achieve better model performance but requires transferring the data to a classic computer for computation, while the latter can be computed by the quantum circuit, e.g. the swap test circuit. Details of loss function design can be found in Appendix D.

### 3.3 Conditional Generation

We further propose a frame-work named QCVAE-Mole, extending upon our above QVAE-Mole by adding certain conditions. Unlike the classic CVAE [35, 36], which achieves conditional generation by simply adding specific condition vectors to input data and latent space. We design to encode conditions into the quantum circuits of the encoder and decoder. Specifically, suppose there is a multi-condition vector $[c_1, c_2, \ldots, c_m]$, where each

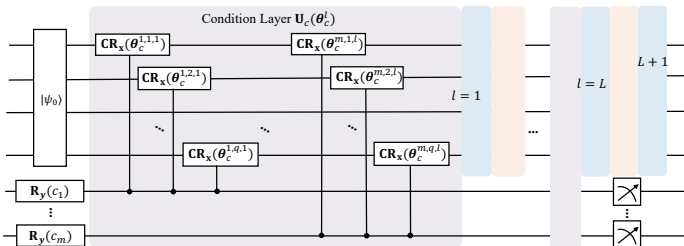

Figure 4: **The ansatz of QCVAE-Mole**: it incorporates $m$ additional condition qubits for $m$ target properties and additional condition layers. At the end of the circuit, we further trace out the state of $m$ condition qubits.

item corresponds to the normalized value (ranging from 0 to 1) of one specific property. Then, we use angle encoding to encode one property into the parameter of one quantum gate, and the initial quantum state becomes:

$$|\Psi_{c_i}\rangle = \mathbf{R_y}\left((c_i - 0.5) \times 2\pi\right) \times |0\rangle. \tag{14}$$

As shown in Fig 4, we extend the scale of the previous quantum circuit and encode the conditions in the last qubits to construct our condition layer. The initial quantum state of the encoder becomes:

$$|\psi_0\rangle^c = |\psi_0\rangle \otimes |\Psi_{c_1}\rangle \otimes \cdots \otimes |\Psi_{c_m}\rangle. \tag{15}$$

Similarly, the input of the decoder undergoes the same change. The unitary matrix of QCVAE is:

$$\mathbf{U}(\boldsymbol{\theta}) = \mathbf{U}_s^{L+1}(\boldsymbol{\theta}) \prod_{l=1}^{L} \left(\mathbf{U}_{ent}^l(\boldsymbol{\theta})\mathbf{U}_s^l(\boldsymbol{\theta})\mathbf{U}_c^l(\boldsymbol{\theta})\right), \tag{16}$$

where the $l$-th condition layer can be formulated as:

$$\mathbf{U}_c^l(\boldsymbol{\theta}) = \prod_{i=1}^{m} \prod_{p=1}^{q} (\mathbf{CR}_x(q + i, p, \boldsymbol{\theta}_c^{(i,p,l)}). \tag{17}$$

Note that every target property requires one more qubit for QCVAE-Mole, which means we have more qubits than QVAE-Mole, but the output dimension of the encoder and decoder should remain the same as vanilla QVAE-Mole, which can be achieved by *tracing out* the condition qubits at the end of the quantum circuit. The process of training is similar to QVAE-Mole and the decoder of the trained model generates molecules with the desired properties according to the given condition vector along with the latent vector.

**Remark.** QCVAE-Mole can be regarded as solving the inverse problem of property prediction, as particularly done in [8]. Indeed, generation could be much more challenging than prediction.

## 4 Experiments

In line with other QML works [8, 19, 20], our experiments are conducted in a simulation environment. Specifically, we use a machine with an i9-10920X CPU, RTX 3090 GPU, and 128G RAM. The source code is written by PyTorch, and TorchQuantum [37] is used as the quantum simulator. In addition, the ansatz of QVAE and QCVAE are all hardware efficient, thus the model can be directly trained on NISQ devices theoretically. Although our approach may involve initial state preparation and quantum state tomography like [15, 29], there are many efficient solutions to address these two challenges, such as [38–42], and this is not the focus of our paper. Detailed discussions of initial state preparation and tomography can be found in Appendix C and F, respectively.

### 4.1 Setting

**Data.** We evaluate QVAE-Mole and QCVAE-Mole on QM9 [43], which is a popular dataset that contains molecular properties and atom coordinates for 130k small molecules with up to 9 heavy atoms. We train our model to either randomly or conditionally generate molecules with 3-D coordinates and atom types (C, N, O, F). We use the train/val partitions introduced in [44], which consist of 100K/18K samples, respectively.

**Actual number of qubits.** When evaluating QM9, we need 7 qubits for random generation, 8 qubits for single-condition generation, and 11 qubits for multiple conditions, respectively.

**Metric.** In line with [23, 45], all the generated 3-D molecules are converted to molecular graphs by the method in [46], and the molecular graphs can be converted to SMILES deterministically with the rdkit [47] toolkit. We use the chemical validity percentage (**Valid**), uniqueness (**Unique**), and novelty (**Novel**) to evaluate the generation quality of QVAE-Mole. Validity measures the percentage of molecules that comply with chemical valency rules, ensuring chemical plausibility. Uniqueness assesses the proportion of distinct molecules generated, promoting structural diversity. Novelty evaluates the fraction of molecules not found in the training data, indicating the model's capacity to generate new compounds. To evaluate the molecular geometry, we use the average Maximum Mean Discrepancy (Avg.MMD) [48] distances of bond length distributions (see details in Appendix G). Note that it is unreasonable to only consider novelty and uniqueness without validity [49]: like in the extreme case if the model's validity is only 1%, but these valid molecules are all unique from each

Table 1: **Comparison of different methods on QM9.** All metrics were evaluated on 10K randomly generated molecules. The method with † denotes it generates molecular graphs, not 3-D molecules, and ∗ denotes it involves quantum computing. #C/Q denotes the number of classic and quantum parameters respectively, and time denotes the inference time to generate one molecule.

| Methods | Model | Class | Valid | Unique×Valid | Novel×Valid | # C/Q Params | Time(s) |
|---|---|---|---|---|---|---|---|
| MLP-VAE [21] | VAE | One-shot | 51.26% $\pm$ 1.1 | 3.43% $\pm$ 0.3 | 48.66% $\pm$ 0.4 | 360,448 / 0 | 0.04 |
| E-NFs [22] | Flow | One-shot | 41.14% | 40.83% | 34.91% | 647,117 / 0 | 0.27 |
| G-SchNet [50] | Sampling | Autoreg. | 82.35% | 73.29% | 67.08% | 902,111 / 0 | 0.41 |
| G-SphereNet [23] | Flow | Autoreg. | 82.63% $\pm$ 1.3 | 29.75% $\pm$ 1.6 | 37.77% $\pm$ 0.9 | 3,148,095 / 0 | 0.55 |
| EDM [49] | Diffusion | One-shot | 91.93% | 90.72% | 75.32% | 5,340,921 / 0 | 0.86 |
| SQ-VAE∗† [15] | VAE | One-shot | 44.23% $\pm$ 1.0 | 7.24% $\pm$ 1.2 | 16.32% $\pm$ 0.8 | 128 / 224 | 0.15 |
| QGAN-HG∗† [14] | GAN | One-shot | 66.64% $\pm$ 0.3 | 8.08% $\pm$ 0.8 | 18.48% $\pm$ 1.0 | 453,644 / 38 | 0.04 |
| P2-QGAN-HG∗† [14] | GAN | One-shot | 17.64% $\pm$ 1.2 | 12.38% $\pm$ 2.1 | 9.54% $\pm$ 0.8 | 112,524 / 14 | 0.02 |
| QVAE-Mole∗ | VAE | One-shot | 78.13% $\pm$ 0.6 | 27.39% $\pm$ 0.8 | 57.38% $\pm$ 0.9 | 0 / 224 | 0.08 |
| QVAE-Mole (fidelity loss)∗ | VAE | One-shot | 74.39% $\pm$ 0.8 | 26.93% $\pm$ 1.0 | 31.50% $\pm$ 0.7 | 0 / 224 | 0.08 |

other and different from the training set, resulting in 100% for both uniqueness and novelty. Thus, we adopt **Unique×Valid** and **Novel×Valid** as metrics.

**Baseline.** We adopt two kinds of methods as our baselines. One category is the classic generation model for 3-D molecules, including MLP-VAE [21], E-NFs [22], G-SchNet [50], G-sphereNet [23] and EDM [49]. Another category contains quantum model SQ-VAE [15] and hybrid model QGAN-HG [14] (P2-QGAN-HG is a variant of QGAN-HG) for molecular graph generation. Note that SQ-VAE still introduces several classic parameters since it needs a linear mapping in latent space. As for QGAN-HG, it simply utilizes quantum circuits to output a feature vector for the classic generator of MOLGAN [51]. To the best of our knowledge, we are the first full quantum model without any classic parameters for 3-D molecule generation.

## 4.2 Random 3-D Molecule Generation

The results are shown in Table 1, reported as the mean with standard deviation across three runs. The training details and results of the MMD distance comparison are shown in Appendix G.

**Compare with classic methods.** The results show that QVAE-Mole surpasses classic MLP-VAE with a notable margin in all metrics, indicating the potential advantages of quantum circuits over classic MLPs. However, there is still a performance gap between our method and the other SOTA baselines, especially EDM, a method based on the diffusion model. On the other hand, when assessing the efficiency of the model, it is also necessary to consider the number of parameters utilized. We can see that our results are very close to the auto-regressive method G-SphereNet and even superior to E-NFs to some extent, which uses 3,148,095 and 647,117 parameters, respectively. In contrast, our model only uses 224 quantum parameters instead. Furthermore, our model has an advantage in inference speed over all the classic models except for the simple VAE. In terms of training cost, our model (even when executed on a simulator) achieves convergence within 2 hours with only a few epochs, which is significantly faster compared to classic methods (according to the original paper on EMD [49], it takes approximately 3.2 days on a 1080Ti GPU to complete 1100 epochs).

**Compare with quantum methods.** Our model outperforms all other quantum or hybrid approaches by a significant margin in all metrics. SQ-VAE uses amplitude encoding for molecular graphs (attributed topology), while the input of our model is a 3-D molecular structure (attributed 3-D point cloud). This indicates that amplitude encoding cannot model topology as effectively as for 3-D point clouds. Furthermore, we utilize a larger latent space compared with SQ-VAE and adopt vMF distribution instead of a normal one, with vMF distribution naturally fitting the inherent and strict normalization requirement of output vectors. The performance of QGAN-HG is poor, probably because the quantum circuits in the hybrid model are unable to fully leverage potential advantages or due to the complex and unstable training process of GAN itself.

## 4.3 Conditional 3-D Molecule Generation

The objective here is to generate molecules with specific desired properties. We train QCVAE-Mole (as discussed in Sec. 3.3) conditioning on four properties: synthetic accessibility score (**SA**),

Table 2: **The results of QCVAE-Mole under single condition given.** Each result is the percentage of the number of molecules that, when rounded up, have the same value as the given condition.

|  | SA ↑ | | QED | | logP | | gap | |
|---|---|---|---|---|---|---|---|---|
| Condition | 0.4 | 0.5 | 0.3 | 0.4 | 0.0 | 1.0 | 3.0 | 4.0 |
| QVAE-Mole | 29.8 | 19.8 | 40.2 | 52.5 | 49.8 | 2.6 | 0.1 | 3.1 |
| QCVAE-Mole | 44.1 | 23.4 | 42.8 | 75.2 | 57.8 | 45.6 | 6.4 | 22.7 |
| $\Delta_{\text{QCVAE-QVAE}}$ | 14.3 | 3.6 | 2.6 | 22.7 | 8.01 | 43.0 | 6.3 | 19.6 |

Comparison of QVAE-Mole and QCVAE-Mole with Multi-condition Generation

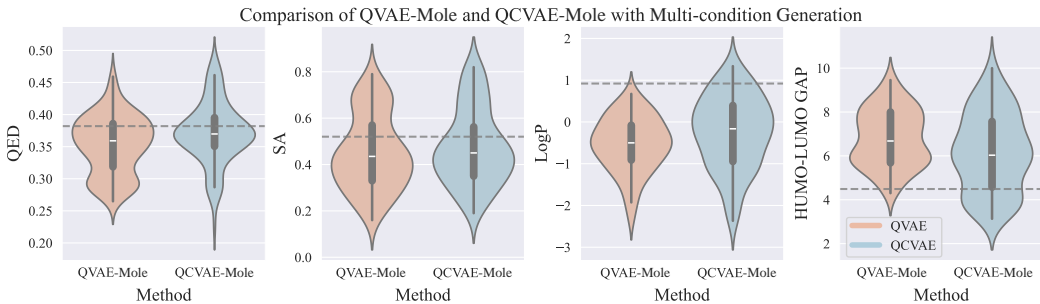

Figure 5: Distribution of four properties of generated molecules **under multi-condition**. Dashed lines represent the given condition values.

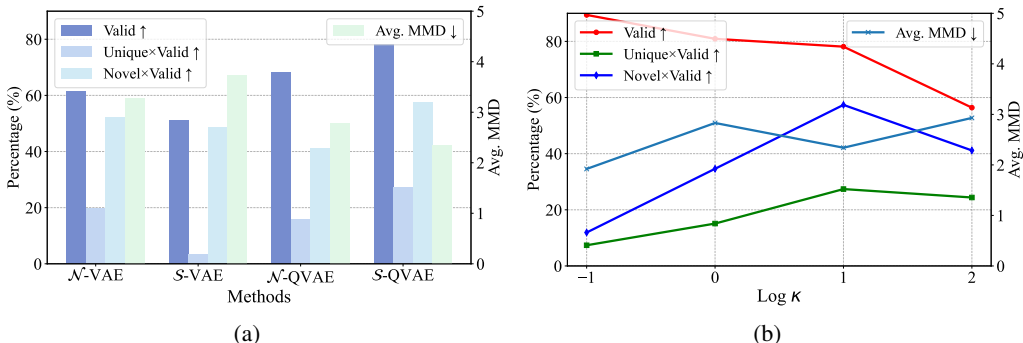

(a)                                                                                        (b)

Figure 6: a) Comparison of a normal distribution ($\mathcal{N}$-) or a vMF one ($\mathcal{S}$-) in the latent space of MLP-VAE (VAE in short) and QVAE-Mole (QVAE in short). b) Different $\kappa$ in vMF distribution.

quantitative estimation of drug-likeness (**QED**), octanol-water partition coefficient (**logP**), and HOMO-LUMO gap (**gap**). See details in Appendix G.

**Single condition.** We train our model with SA, QED, logP, and gap, respectively, resulting in four models. For each model, we compare the conditional generation (QCVAE-Mole) with random generation (QVAE-Mole) and the corresponding results are shown in Table 2. Each column in Table 2 indicates the percentage of molecules whose properties, when rounded up, match the specified condition. For instance, for the condition logP = 0.0, the results show that 57.8% of generated molecules have logP values within the range [-0.5, 0.5]. The results demonstrate that QCVAE-Mole can improve the proportion of generated molecules with desired properties for all given conditions, the improvement even reaches 43% when we give the condition with logP= 1.

**Multiple conditions.** We train the proposed QCVAE-Mole under multiple conditions, which means we give SA, QED, logP, and gap these four properties simultaneously. To evaluate under a combination of multiple conditions, following [52], we select a reference molecule and adopt its property values as our multi-condition. Here we choose "CC1=C=C=C(N=O)C1=O", and the property values are: $\{\text{SA} = 0.52, \text{QED} = 0.38, \text{logP} = 0.92, \text{gap} = 4.45\}$. Fig. 5 illustrates the comparison with random generation, demonstrating that QCVAE-Mole is capable of achieving multi-condition generation, albeit with less pronounced improvement compared to single-condition generation.

## 4.4 Ablation Study

**Normal distribution vs vMF distribution.** We study adopting normal distribution and von Mises-Fisher (vMF) distribution in classic VAE and quantum VAE, corresponding to Euclidean space and hyperspherical space, respectively. Here we conduct experiments under random generation setting. Fig. 6(a) shows QVAE with hyperspherical latent space performs the best validity, which reveals the advantages of using vMF distribution in QVAE. In addition, we found that compared to $\mathcal{N}$-VAE, $\mathcal{N}$-QVAE can generate relatively reasonable 3-D coordinates, ensuring that the distances between atoms fall within the range of chemical bond lengths. However, the atom numbers and atom types generated by $\mathcal{N}$-QVAE are relatively homogeneous, resulting in lower scores for novel and unique compared to $\mathcal{N}$-VAE. We also observe that utilizing normal distribution leads to better performance in classic VAE than using the von Mises-Fisher distribution.

**Compact of $\kappa$.** In Sec. 3.2.2, we have mentioned the concentration parameter $\kappa$ in vMF distribution is commonly set as a constant during training. To further explore its impact on the quality of random generation, here we vary $\kappa$ from 0.1 to 100 and the results are shown in Fig. 6(b). We observe that the smaller the $\kappa$ value, the higher the validity, but the uniqueness and novelty will decrease correspondingly. This indicates a trade-off in the selection of $\kappa$: when we need high accuracy, we should choose a smaller $\kappa$, while diverse molecules are in need, we should opt for a larger $\kappa$.

## 5 Conclusion, Limitation, and Outlook

We have proposed the first fully quantum VAE for 3-D molecule generation, to the best of our knowledge, featuring a von Mises-Fisher distributed latent space. Moreover, we have designed a conditional version for target molecule generation. Numerical experiments show that our model could generate plausible 3-D molecules, which outperform all other quantum (or hybrid) methods, and achieve competitive performance with significantly reduced parameters compared with their classic counterparts. Though we provide detailed quantum circuits compatible with NISQ devices, due to hardware constraints, so far we have not implemented our proposed quantum circuits on real quantum devices. We leave it for future work and further collaboration with hardware groups where tailored hardware error correction may be needed.

## References

[1] P. Schneider, W. P. Walters, A. T. Plowright, N. Sieroka, J. Listgarten, R. A. Goodnow Jr, J. Fisher, J. M. Jansen, J. S. Duca, T. S. Rush *et al.*, "Rethinking drug design in the artificial intelligence era," *Nature Reviews Drug Discovery*, vol. 19, no. 5, pp. 353–364, 2020.

[2] L. Patel, T. Shukla, X. Huang, D. W. Ussery, and S. Wang, "Machine learning methods in drug discovery," *Molecules*, vol. 25, no. 22, p. 5277, 2020.

[3] D. Paul, G. Sanap, S. Shenoy, D. Kalyane, K. Kalia, and R. K. Tekade, "Artificial intelligence in drug discovery and development," *Drug discovery today*, vol. 26, no. 1, p. 80, 2021.

[4] S. Khairy, R. Shaydulin, L. Cincio, Y. Alexeev, and P. Balaprakash, "Learning to optimize variational quantum circuits to solve combinatorial problems," in *Proceedings of the AAAI conference on artificial intelligence*, vol. 34, no. 03, 2020, pp. 2367–2375.

[5] Y. Du, M.-H. Hsieh, T. Liu, D. Tao, and N. Liu, "Quantum noise protects quantum classifiers against adversaries," *Physical Review Research*, vol. 3, no. 2, p. 023153, 2021.

[6] N. S. Blunt, J. Camps, O. Crawford, R. Izsák, S. Leontica, A. Mirani, A. E. Moylett, S. A. Scivier, C. Sunderhauf, P. Schopf *et al.*, "Perspective on the current state-of-the-art of quantum computing for drug discovery applications," *Journal of Chemical Theory and Computation*, vol. 18, no. 12, pp. 7001–7023, 2022.

[7] S. McArdle, S. Endo, A. Aspuru-Guzik, S. C. Benjamin, and X. Yuan, "Quantum computational chemistry," *Reviews of Modern Physics*, vol. 92, no. 1, p. 015003, 2020.

[8] G. Yan, H. Wu, and J. Yan, "Quantum 3d graph learning with applications to molecule embedding," in *Proceedings of the International Conference on Machine Learning*. PMLR, 2023, pp. 39 126–39 137.

[9] R. Huang, X. Tan, and Q. Xu, "Learning to learn variational quantum algorithm," *IEEE Transactions on Neural Networks and Learning Systems*, 2022.

[10] K. Beer, D. Bondarenko, T. Farrelly, T. J. Osborne, R. Salzmann, D. Scheiermann, and R. Wolf, "Training deep quantum neural networks," *Nature communications*, vol. 11, no. 1, p. 808, 2020.

[11] A. Abbas, D. Sutter, C. Zoufal, A. Lucchi, A. Figalli, and S. Woerner, "The power of quantum neural networks," *Nature Computational Science*, vol. 1, no. 6, pp. 403–409, 2021.

[12] H.-L. Huang, Y. Du, M. Gong, Y. Zhao, Y. Wu, C. Wang, S. Li, F. Liang, J. Lin, Y. Xu *et al.*, "Experimental quantum generative adversarial networks for image generation," *Physical Review Applied*, vol. 16, no. 2, p. 024051, 2021.

[13] X. Ye, G. Yan, and J. Yan, "Towards quantum machine learning for constrained combinatorial optimization: a quantum qap solver," in *Proceedings of the International Conference on Machine Learning*. PMLR, 2023, pp. 39 903–39 912.

[14] J. Li, R. O. Topaloglu, and S. Ghosh, "Quantum generative models for small molecule drug discovery," *IEEE transactions on quantum engineering*, vol. 2, pp. 1–8, 2021.

[15] J. Li and S. Ghosh, "Scalable variational quantum circuits for autoencoder-based drug discovery," in *2022 Design, Automation & Test in Europe Conference & Exhibition*. IEEE, 2022, pp. 340–345.

[16] J. Zhou, G. Cui, S. Hu, Z. Zhang, C. Yang, Z. Liu, L. Wang, C. Li, and M. Sun, "Graph neural networks: A review of methods and applications," *AI open*, vol. 1, pp. 57–81, 2020.

[17] K. Qiu, Y. Chen, T. W. Rees, L. Ji, and H. Chao, "Organelle-targeting metal complexes: From molecular design to bio-applications," *Coordination Chemistry Reviews*, vol. 378, pp. 66–86, 2019.

[18] A. Mishra and P. Bäuerle, "Small molecule organic semiconductors on the move: promises for future solar energy technology," *Angewandte Chemie International Edition*, vol. 51, no. 9, pp. 2020–2067, 2012.

[19] J. Bausch, "Recurrent quantum neural networks," *Proceedings of the Advances in neural information processing systems*, vol. 33, pp. 1368–1379, 2020.

[20] X. Ye, G. Yan, and J. Yan, "Vqne: Variational quantum network embedding with application to network alignment," in *Proceedings of the ACM SIGKDD Conference on Knowledge Discovery and Data Mining*, 2023, pp. 3105–3115.

[21] D. P. Kingma, "Auto-encoding variational bayes," *arXiv preprint arXiv:1312.6114*, 2013.

[22] V. Garcia Satorras, E. Hoogeboom, F. Fuchs, I. Posner, and M. Welling, "E (n) equivariant normalizing flows," *Proceedings of the Advances in Neural Information Processing Systems*, vol. 34, pp. 4181–4192, 2021.

[23] Y. Luo and S. Ji, "An autoregressive flow model for 3d molecular geometry generation from scratch," in *Proceedings of the International Conference on Learning Representations*, 2022.

[24] M. Cerezo, A. Arrasmith, R. Babbush, S. C. Benjamin, S. Endo, K. Fujii, J. R. McClean, K. Mitarai, X. Yuan, L. Cincio *et al.*, "Variational quantum algorithms," *Nature Reviews Physics*, vol. 3, no. 9, pp. 625–644, 2021.

[25] K. Mitarai, M. Negoro, M. Kitagawa, and K. Fujii, "Quantum circuit learning," *Physical Review A*, vol. 98, no. 3, p. 032309, 2018.

[26] A. Senokosov, A. Sedykh, A. Sagingalieva, B. Kyriacou, and A. Melnikov, "Quantum machine learning for image classification," *Machine Learning: Science and Technology*, 2023.

[27] I. Cong, S. Choi, and M. D. Lukin, "Quantum convolutional neural networks," *Nature Physics*, vol. 15, no. 12, pp. 1273–1278, 2019.

[28] S. Y.-C. Chen, S. Yoo, and Y.-L. L. Fang, "Quantum long short-term memory," in *Proceedings of the IEEE International Conference on Acoustics, Speech and Signal Processing*. IEEE, 2022, pp. 8622–8626.

[29] L. Rathi, E. Tretschk, C. Theobalt, R. Dabral, and V. Golyanik, "3d-qae: Fully quantum auto-encoding of 3d point clouds," *arXiv preprint arXiv:2311.05604*, 2023.

[30] J. Romero, J. P. Olson, and A. Aspuru-Guzik, "Quantum autoencoders for efficient compression of quantum data," *Quantum Science and Technology*, vol. 2, no. 4, p. 045001, 2017.

[31] G. M. D'Ariano, M. G. Paris, and M. F. Sacchi, "Quantum tomography," *Advances in imaging and electron physics*, vol. 128, pp. 206–309, 2003.

[32] J. Xu and G. Durrett, "Spherical latent spaces for stable variational autoencoders," *arXiv preprint arXiv:1808.10805*, 2018.

[33] T. R. Davidson, L. Falorsi, N. De Cao, T. Kipf, and J. M. Tomczak, "Hyperspherical variational auto-encoders," in *Proceedings of the Conference on Uncertainty in Artificial Intelligence*. Association For Uncertainty in Artificial Intelligence, 2018, pp. 856–865.

[34] S. R. Bowman, L. Vilnis, O. Vinyals, A. M. Dai, R. Jozefowicz, and S. Bengio, "Generating sentences from a continuous space," *arXiv preprint arXiv:1511.06349*, 2015.

[35] M. Lee and K. Min, "Mgcvae: multi-objective inverse design via molecular graph conditional variational autoencoder," *Journal of Chemical Information and Modeling*, vol. 62, no. 12, pp. 2943–2950, 2022.

[36] A. Mishra, S. Krishna Reddy, A. Mittal, and H. A. Murthy, "A generative model for zero shot learning using conditional variational autoencoders," in *Proceedings of the IEEE conference on computer vision and pattern recognition workshops*, 2018, pp. 2188–2196.

[37] H. Wang, Y. Ding, J. Gu, Y. Lin, D. Z. Pan, F. T. Chong, and S. Han, "Quantumnas: Noise-adaptive search for robust quantum circuits," in *Proceedings of the IEEE International Symposium on High-Performance Computer Architecture*. IEEE, 2022, pp. 692–708.

[38] M. Weigold, J. Barzen, F. Leymann, and M. Salm, "Data encoding patterns for quantum computing," in *Proceedings of the 27th Conference on Pattern Languages of Programs*, 2020, pp. 1–11.

[39] I. F. Araujo, D. K. Park, T. B. Ludermir, W. R. Oliveira, F. Petruccione, and A. J. Da Silva, "Configurable sublinear circuits for quantum state preparation," *Quantum Information Processing*, vol. 22, no. 2, p. 123, 2023.

[40] T. Schmale, M. Reh, and M. Gärttner, "Efficient quantum state tomography with convolutional neural networks," *npj Quantum Information*, vol. 8, no. 1, p. 115, 2022.

[41] A. W. Smith, J. Gray, and M. Kim, "Efficient quantum state sample tomography with basis-dependent neural networks," *PRX Quantum*, vol. 2, no. 2, p. 020348, 2021.

[42] M. Rambach, M. Qaryan, M. Kewming, C. Ferrie, A. G. White, and J. Romero, "Robust and efficient high-dimensional quantum state tomography," *Physical Review Letters*, vol. 126, no. 10, p. 100402, 2021.

[43] R. Ramakrishnan, P. O. Dral, M. Rupp, and O. A. Von Lilienfeld, "Quantum chemistry structures and properties of 134 kilo molecules," *Scientific data*, vol. 1, no. 1, pp. 1–7, 2014.

[44] B. Anderson, T. S. Hy, and R. Kondor, "Cormorant: Covariant molecular neural networks," *Proceedings of the Advances in neural information processing systems*, vol. 32, 2019.

[45] M. Liu, Y. Luo, K. Uchino, K. Maruhashi, and S. Ji, "Generating 3d molecules for target protein binding," in *Proceedings of the International Conference on Machine Learning*, 2022.

[46] Y. Kim and W. Y. Kim, "Universal structure conversion method for organic molecules: from atomic connectivity to three-dimensional geometry," *Bulletin of the Korean Chemical Society*, vol. 36, no. 7, pp. 1769–1777, 2015.

[47] G. Landrum *et al.*, "Rdkit: A software suite for cheminformatics, computational chemistry, and predictive modeling," *Greg Landrum*, vol. 8, p. 31, 2013.

[48] A. Gretton, K. M. Borgwardt, M. J. Rasch, B. Schölkopf, and A. Smola, "A kernel two-sample test," *The Journal of Machine Learning Research*, vol. 13, no. 1, pp. 723–773, 2012.

[49] E. Hoogeboom, V. G. Satorras, C. Vignac, and M. Welling, "Equivariant diffusion for molecule generation in 3d," in *Proceedings of the International conference on machine learning*. PMLR, 2022, pp. 8867–8887.

[50] N. Gebauer, M. Gastegger, and K. Schütt, "Symmetry-adapted generation of 3d point sets for the targeted discovery of molecules," *Proceedings of the Advances in neural information processing systems*, vol. 32, 2019.

[51] N. De Cao and T. Kipf, "MolGAN: An implicit generative model for small molecular graphs," *Proceedings of the ICML 2018 workshop on Theoretical Foundations and Applications of Deep Generative Models*, 2018.

[52] J. Lim, S. Ryu, J. W. Kim, and W. Y. Kim, "Molecular generative model based on conditional variational autoencoder for de novo molecular design," *Journal of cheminformatics*, vol. 10, no. 1, pp. 1–9, 2018.

[53] M. Simonovsky and N. Komodakis, "Graphvae: Towards generation of small graphs using variational autoencoders," in *Artificial Neural Networks and Machine Learning–ICANN 2018: 27th International Conference on Artificial Neural Networks, Rhodes, Greece, October 4-7, 2018, Proceedings, Part I 27*. Springer, 2018, pp. 412–422.

[54] J. Jo, S. Lee, and S. J. Hwang, "Score-based generative modeling of graphs via the system of stochastic differential equations," in *Proceedings of the International Conference on Machine Learning*. PMLR, 2022, pp. 10 362–10 383.

[55] C. Vignac, I. Krawczuk, A. Siraudin, B. Wang, V. Cevher, and P. Frossard, "Digress: Discrete denoising diffusion for graph generation," in *Proceedings of the International Conference on Learning Representations*, 2023.

[56] W. Jin, R. Barzilay, and T. Jaakkola, "Junction tree variational autoencoder for molecular graph generation," in *Proceedings of the International Conference on Machine Learning*. PMLR, 2018, pp. 2323–2332.

[57] H. Kajino, "Molecular hypergraph grammar with its application to molecular optimization," in *Proceedings of the International Conference on Machine Learning*. PMLR, 2019, pp. 3183–3191.

[58] X. Kong, W. Huang, Z. Tan, and Y. Liu, "Molecule generation by principal subgraph mining and assembling," *Proceedings of the Advances in Neural Information Processing Systems*, vol. 35, pp. 2550–2563, 2022.

[59] Y. Xie, C. Shi, H. Zhou, Y. Yang, W. Zhang, Y. Yu, and L. Li, "{MARS}: Markov molecular sampling for multi-objective drug discovery," in *Proceedings of the International Conference on Learning Representations*, 2021.

[60] C. Shi, M. Xu, Z. Zhu, W. Zhang, M. Zhang, and J. Tang, "Graphaf: a flow-based autoregressive model for molecular graph generation," in *Proceedings of the International Conference on Learning Representations*, 2020.

[61] Y. Luo, K. Yan, and S. Ji, "Graphdf: A discrete flow model for molecular graph generation," in *Proceedings of the International Conference on Machine Learning*. PMLR, 2021, pp. 7192–7203.

[62] S. Ahn, B. Chen, T. Wang, and L. Song, "Spanning tree-based graph generation for molecules," in *Proceedings of the International Conference on Learning Representations*, 2022.

[63] J. Ho, A. Jain, and P. Abbeel, "Denoising diffusion probabilistic models," *Proceedings of the Advances in neural information processing systems*, vol. 33, pp. 6840–6851, 2020.

[64] S. Zhang, J. Huang, Q. Zhou, Z. Wang, F. Wang, J. Luo, and J. Yan, "Continuous-multiple image outpainting in one-step via positional query and a diffusion-based approach," *Proceedings of the International Conference on Learning Representations*, 2024.

[65] S. Zhang, M. Liu, J. Yan, H. Zhang, L. Huang, X. Yang, and P. Lu, "M-mix: Generating hard negatives via multi-sample mixing for contrastive learning," in *Proceedings of the ACM SIGKDD Conference on Knowledge Discovery and Data Mining*, 2022.

[66] L. Wu, C. Gong, X. Liu, M. Ye, and Q. Liu, "Diffusion-based molecule generation with informative prior bridges," *Proceedings of the Advances in Neural Information Processing Systems*, vol. 35, pp. 36 533–36 545, 2022.

[67] M. Benedetti, D. Garcia-Pintos, O. Perdomo, V. Leyton-Ortega, Y. Nam, and A. Perdomo-Ortiz, "A generative modeling approach for benchmarking and training shallow quantum circuits," *npj Quantum Information*, vol. 5, no. 1, p. 45, 2019.

[68] D. Zhu, N. M. Linke, M. Benedetti, K. A. Landsman, N. H. Nguyen, C. H. Alderete, A. Perdomo-Ortiz, N. Korda, A. Garfoot, C. Brecque *et al.*, "Training of quantum circuits on a hybrid quantum computer," *Science advances*, vol. 5, no. 10, p. eaaw9918, 2019.

[69] M. H. Amin, E. Andriyash, J. Rolfe, B. Kulchytskyy, and R. Melko, "Quantum boltzmann machine," *Physical Review X*, vol. 8, no. 2, p. 021050, 2018.

[70] S. Lloyd and C. Weedbrook, "Quantum generative adversarial learning," *Physical review letters*, vol. 121, no. 4, p. 040502, 2018.

[71] A. Khoshaman, W. Vinci, B. Denis, E. Andriyash, H. Sadeghi, and M. H. Amin, "Quantum variational autoencoder," *Quantum Science and Technology*, vol. 4, no. 1, p. 014001, 2018.

[72] N. Gao, M. Wilson, T. Vandal, W. Vinci, R. Nemani, and E. Rieffel, "High-dimensional similarity search with quantum-assisted variational autoencoder," in *Proceedings of the ACM SIGKDD international conference on knowledge discovery data mining*, 2020, pp. 956–964.

[73] X. Zhang, S. Zhang, and J. Yan, "Pcp-mae: Learning to predict centers for point masked autoencoders," *Proceedings of the Advances in Neural Information Processing Systems*, 2024.

[74] M. Schuld, "Quantum machine learning models are kernel methods (2021)," *arXiv preprint arXiv:2101.11020*.

[75] L. Bai, L. Rossi, L. Cui, Z. Zhang, P. Ren, X. Bai, and E. Hancock, "Quantum kernels for unattributed graphs using discrete-time quantum walks," *Pattern Recognition Letters*, vol. 87, pp. 96–103, 2017.

[76] M. S. Rudolph, J. Chen, J. Miller, A. Acharya, and A. Perdomo-Ortiz, "Decomposition of matrix product states into shallow quantum circuits," *Quantum Science and Technology*, vol. 9, no. 1, p. 015012, 2023.

[77] W. Wu, G. Yan, X. Lu, K. Pan, and J. Yan, "Quantumdarts: differentiable quantum architecture search for variational quantum algorithms," in *Proceedings of the International Conference on Machine Learning*. PMLR, 2023, pp. 37 745–37 764.

[78] A. Barenco, C. H. Bennett, R. Cleve, D. P. DiVincenzo, N. Margolus, P. Shor, T. Sleator, J. A. Smolin, and H. Weinfurter, "Elementary gates for quantum computation," *Physical review A*, vol. 52, no. 5, p. 3457, 1995.

[79] A. K. Ghose, V. N. Viswanadhan, and J. J. Wendoloski, "A knowledge-based approach in designing combinatorial or medicinal chemistry libraries for drug discovery. 1. a qualitative and quantitative characterization of known drug databases," *Journal of combinatorial chemistry*, vol. 1, no. 1, pp. 55–68, 1999.

[80] D. P. Kingma and J. Ba, "Adam: A method for stochastic optimization," *arXiv preprint arXiv:1412.6980*, 2014.

# Appendix

## A Related Work

**Classic methods for molecule generation.** Classic deep learning for molecule generation can be roughly divided into two categories. The first category focuses on generating molecular graphs, which has been extensively studied in the research community. These methods include generating the node type and adjacency matrix of the graph all at once [51, 53–55], or generating molecular graphs using fragments or motifs [56–59], or sequentially adding nodes and edges to molecular graph [60–62]. Another category focuses on a relatively under-investigated domain, which involves generating the 3-D molecular structure. G-SchNet [50] and G-SphereNet [23] use auto-regressive models to generate molecules in a step-by-step manner via progressively connecting atoms. E-NF [22] introduces an equivariant normalizing flow that incorporates a differential equation. Recently, with the emergence and success of diffusion models (DM) [63–65], there has been a notable shift towards utilizing DMs for 3D molecule generation [49, 66].

**Quantum generative learning.** There are four types of quantum generative models: Quantum Circuit Born Machines (QCBMs) [67, 68], Quantum Boltzmann Machines (QBMs) [69], Quantum Generative Adversarial Networks (QGANs) [70, 12], and Quantum Variational Autoencoders (QVAEs) [71, 72]. The complexity of QCBM-based and QBM-based methods is high, which makes them difficult to implement on NISQ devices. QGAN conceptualized by [70] discusses the potential merits of QGANs when either the generator, the discriminator, or both are implemented on quantum computers. However, balancing the training rates of quantum generators and discriminators remains a critical challenge. The QVAE introduced by [71] adopts a classic VAE structure [21, 73] and a quantum prior distribution in the latent space realized by a QBM model. However, it remains an open question whether there is a quantum circuit realization of QBM. In contrast, we explicitly present a full quantum VAE circuit. As for quantum generative learning for molecule design, QGAN-HG [14] introduces a hybrid model upon MOLGAN, and SQ-VAE [15] proposes a circuit-based QVAE, which is more hardware efficient compared to QVAE based on QBM model. In this paper, we follow the paradigm of circuit-based quantum VAE and propose a fully quantum VAE for 3-D molecule generation with the capacity of conditional generation. Although SQ-VAE is also designed for molecular generation, our methodological framework is fundamentally different. Specifically, our input and output are tailored for 3D molecular structures, reflected in the encoding of input information. And the sampling of latent space variables and the final measurement method of the quantum circuit are also different. Moreover, we propose a full quantum neural network capable of multi-conditional control as encoder/decoder while SQ-VAE uses a hybrid quantum-classical layer. In defining the latent variable space, we employ the vMF distribution to harness the inherent properties of quantum states, while they simply imitate a classical VAE using a Gaussian distribution.

## B Quantum Computing and Parameterized Quantum Gates

Quantum states are typically represented using bracket notation. It is common practice to create linear combinations of these states, known as a superposition, exemplified by $|\psi\rangle = \alpha|0\rangle + \beta|1\rangle$. In formal terms, a quantum system composed of $n$ qubits is represented by an $n$-fold tensor product Hilbert space $\mathcal{H} = \left(\mathbb{C}^2\right)^{\otimes d}$, which has a dimension of $2^d$. For any state $|\psi\rangle$ within $\mathcal{H}$, its conjugate transpose is denoted as $\langle\psi| = |\psi\rangle^\dagger$. The inner product $\langle\psi|\psi\rangle = |\psi|_2^2$ calculates the square of the 2-norm of $\psi$. The outer product $|\psi\rangle\langle\psi|$ forms a rank 2 tensor. The computational basis states are defined as $|0\rangle = (1,0)$ and $|1\rangle = (0,1)$, while composite basis states, such as $|01\rangle = |0\rangle \otimes |1\rangle = (0,1,0,0)$, extend these definitions.

The single-qubit and double-qubit parameterized quantum gates used in this paper contain:

$$\mathbf{R_y}(\theta) = \begin{bmatrix} \cos(\frac{\theta}{2}) & -\sin(\frac{\theta}{2}) \\ \sin(\frac{\theta}{2}) & \cos(\frac{\theta}{2}) \end{bmatrix}, \mathbf{R_z}(\theta) = \begin{bmatrix} e^{-i\frac{\theta}{2}} & 0 \\ 0 & e^{i\frac{\theta}{2}} \end{bmatrix},$$

$$\mathbf{CR_x}(\theta) = \begin{bmatrix} 1 & 0 & 0 & 0 \\ 0 & 1 & 0 & 0 \\ 0 & 0 & \cos(\frac{\theta}{2}) & -i\sin(\frac{\theta}{2}) \\ 0 & 0 & -i\sin(\frac{\theta}{2}) & \cos(\frac{\theta}{2}) \end{bmatrix}.$$

Similar to a classical computer, a quantum computer is constructed using a quantum circuit composed of wires and basic quantum gates, which facilitate the transport and manipulation of quantum information. Each quantum gate represents a unitary operation, denoted as U, on a Hilbert space $\mathcal{H}$. When simulating a quantum circuit on a classical computer, the complete unitary transformation is achieved by tensoring and multiplying these unitary gate operators together.

A projective measurement involves an observable, denoted as $M$, which is a Hermitian operator acting on the state space of the observed system. This observable can be broken down into a spectral decomposition, represented as $M = \sum_m m\mathbf{P}_m$, where each $\mathbf{P}_m$ is the projector onto the eigenspace of $M$ corresponding to the eigenvalue $m$. When the state $|\psi\rangle$ is measured, the probability of obtaining the result $m$ is determined by $p(m) = \langle\psi|\mathbf{P}_m|\psi\rangle$.

## C  Encoding Details

The dataset is a set of $M$ molecules, which we represent as $M$ 3-D point clouds, each with $n_j$ nodes. To ensure that molecules can be uniquely encoded to a quantum state vector, we need to fix the order of atoms and the molecular conformation during input. Furthermore, the probability vector containing the amplitudes restricts the generated vector to the positive octant, which requires us to keep the input data of molecule within the positive octant as well.

To fix the order of atoms, we first convert the 3-D molecules to canonical SMILES strings using rdkit. We then encode the coordinates and atom type of each atom onto the quantum state vector according to the canonical SMILES strings. As for the molecular conformation, we first align the center of mass of the entire molecule to zero and then rotate the position of the first atom in the SMILES string onto the $z$-axis. Now molecules in dataset with arbitrary conformations and atom orders can be uniquely represented by $[\{(\mathbf{v}_i^j, \mathbf{a}_i^j)\}_{i=0}^{n_j-1}]_{j=0}^{M-1}$.

In addition, to keep the molecule within the octant, we define an axis-aligned bounding box $(\mathbf{v}_{min}, \mathbf{v}_{max})$ across the entire dataset. This is done by defining the minimum and maximum values along each axis.

$$v_{\min,a} = \min_{\substack{j=0,\ldots,M-1 \\ i=0,\ldots,n_j-1}} v_{i,a}^j, \qquad v_{\max,a} = \max_{\substack{j=0,\ldots,M-1 \\ i=0,\ldots,n_j-1}} v_{i,a}^j,$$

where $v_{i,a}^j \in \mathbb{R}$ is the coordinate of vertex $v_i^j$ along axis $a \in \{x,y,z\}$. To further achieve isotropic re-scaling, we turn the bounding box into a cube with side length $s = \max_{a \in x,y,z} v_{\max,a} - v_{\min,a}$. Then the coordinates of each atom can be shifted and re-scaled by

$$[\{\tilde{\mathbf{v}}_i^j\}_{i=0}^{n_i-1}]_{j=0}^{M-1} = [\{\frac{\mathbf{v}_i^j - \mathbf{v}_{\min}}{s}\}_{i=0}^{n_j-1}]_{j=0}^{M-1}$$

Now we get the final normalized dataset.

**Initial state preparation.** There are several techniques for encoding classical data into quantum states, including basis encoding, angle encoding, and amplitude encoding [74]. We adopt the amplitude encoding scheme to encode the input 3-D molecule or sampled classical latent variable as a quantum state. The reason we adopt amplitude encoding is that it can encode $2^n$ classical values using only $n$ qubits, offering an exponential encoding advantage. In the existing literature on quantum algorithms, amplitude encoding is commonly employed to harness the power of quantum computation [75]. The implementation of amplitude encoding is complex, but there are many works dedicated to investigating how to efficiently encode information into amplitudes [38], such as top-down encoding [39], bottom-top encoding [39], MPS encoding [76], or quantum architecture search [77].

**The qubits complexity analysis of QGAN-HG and SQ-VAE.** QGAN-HG [14] is a hybrid model with a classic discriminator and a hybrid generator, which utilize quantum circuits to generate feature vectors. QGAN-HG obtains the output of the circuit through measurements and yields a $q$-bit string (here $q$ is the number of qubits). Thus the qubits complexity comes to $q = d = \mathcal{O}(d)$, where $d$ is the needed feature dimension.

SQ-VAE [15] uses amplitude encoding to encode molecular graphs. First, the molecular graph is represented by a node matrix ($n \times k$) and bond matrix ($n \times n \times b$), $n$ is the number of atoms, $k$

is the number of atom types and $b$ is the bond types. Then, the node matrix and bond matrix are flattened to get a vector with the dimension of $n * k + n * n * k$. Thus the qubits complexity comes to $q = \lceil \log_2(n * k + n * n * k) \rceil = \mathcal{O}(C \log(n^2)) = \mathcal{O}(C' \log(n))$

## D  Loss Function Design

**Classic loss.** The classic reconstruction loss consists of four parts: 3-D coordinate loss $\mathcal{L}_1$, atomic classification loss $\mathcal{L}_2$, constraint loss $\mathcal{L}_3$, and auxiliary loss $\mathcal{L}_4$. We use *geometric distance error* for 3-D coordinate loss, *weighted cross entropy* for atomic classification loss with weight $w_t$ and one-hot label $y_t$ for each atom type $t$. Note that the sum of the probability of each atom type $s_i = \alpha_{r+3}^2 + \ldots + \alpha_{r+3+(k-1)}^2$ should be as equal as possible for each $i$, with the expectation of $\mathbb{E}[s_i] = (\frac{1}{2\sqrt{n}})^2 = \frac{1}{4n}$. To this end, we design a constraint loss via *mean square error*. Furthermore, the auxiliary loss is designed for the auxiliary value and the padding entries. In summary:

$$\mathcal{L}_1 = \sum_i \left( \left( \alpha_r - \frac{x_i}{2\sqrt{n}} \right)^2 + \left( \alpha_{r+1} - \frac{y_i}{2\sqrt{n}} \right)^2 + \left( \alpha_{r+2} - \frac{z_i}{2\sqrt{n}} \right)^2 \right)^{1/2}, \tag{18}$$

$$\mathcal{L}_2 = -\sum_i \sum_t w_t y_{it} \log(\alpha_{it}), \quad \mathcal{L}_3 = \sum_i (s_i - \frac{1}{4n})^2, \tag{19}$$

$$\mathcal{L}_4 = \sum_i \left| \alpha_{r+k+3} - \frac{\sqrt{3 - x_i^2 - y_i^2 - z_i^2}}{2\sqrt{n}} \right| + \sum_{j=(n*(4+k))}^{2^q-1} |\alpha_j|. \tag{20}$$

Specifically, $L_1$ supervises the reconstruction of the molecule 3-D position by geometric distance error, and $L_2$ supervises the reconstruction of atom types by weighted cross entropy. $L_3$ is used to constrain the sum of the probability of all atom types for each atom to be the same, and we use MSE loss here. Since we add padding entries to the input data, so $L_4$ is designed to supervise the reconstruction of these entries of zero. With $M$ data samples in total, the final loss comes to:

$$\mathcal{L} = \frac{1}{M} \sum (\mathcal{L}_1 + \alpha \mathcal{L}_2 + \beta \mathcal{L}_3 + \gamma \mathcal{L}_4), \tag{21}$$

where $\alpha, \beta, \gamma$ are the hyperparameters to balance the losses. The best hyperparameter configuration can be determined through grid search.

**Fidelity loss.** It quantifies the "closeness" between two quantum states, and the formal definition between two quantum states $\rho, \sigma$ is $F(\rho, \sigma) = (\text{tr}\sqrt{\sqrt{\rho}\sigma\sqrt{\rho}})^2$. For pure states, $\rho = |\psi_\rho\rangle\langle\psi_\rho|$ and $\sigma = |\psi_\sigma\rangle\langle\psi_\sigma|$, the fidelity equals to $\langle\psi_\rho|\psi_\sigma\rangle$. We construct the fidelity loss:

$$\mathcal{L}_f = 1 - F(\psi_0, \psi_r) = 1 - |\langle\psi_0 \mid \psi_r\rangle|^2. \tag{22}$$

The design of fidelity loss does not consider the real physical meaning of the output quantum vector, but instead treats the input and decoder output of the encoder as two quantum states, and then designs the loss by calculating the fidelity between them.

## E  Discussion of Circuit Design

We choose $\mathbf{R_z}(\theta)\mathbf{R_y}(\theta)\mathbf{R_z}(\theta)$ as the single-qubit trainable layer, and next we will introduce the reason. An arbitrary single-qubit operator with the unitary matrix as $U$ can be decomposed into a sequence of $\mathbf{R_z}$, $\mathbf{R_y}$, and $\mathbf{R_z}$ gates, and a global phase [78].

$$U = e^{i\alpha}\mathbf{R_z}(\theta_2)\mathbf{R_y}(\theta_1)\mathbf{R_z}(\theta_0) \tag{23}$$

*Proof.* A $2 \times 2$ unitary matrix has the following general expression:

$$U = e^{i\alpha} \begin{bmatrix} a & -b^* \\ b & a^* \end{bmatrix} = e^{i\alpha}V, \tag{24}$$

where $a, b$ are complex numbers, and $\alpha$ is a real number, $e^{i\alpha}$ is the phase part, and $V$ is also a unitary matrix. Notice that the determinant of $V$ satisfies $\det V = aa^* + bb^* = |a|^2 + |b|^2 = 1$ (* denotes the conjugate operator). Consequently, we have

$$\det U = e^{2i\alpha} \det V = e^{2i\alpha}. \tag{25}$$

---

**Algorithm 1** vMF sampling

---

**Input**: dimension $m$, direction mean $\mu$, constant $\kappa$.

1: **sample v** $\sim U\left(\mathcal{S}^{m-2}\right)$;

2: **sample** $\omega \sim g(\omega \mid \kappa, m) \propto \exp(\omega\kappa)\left(1-\omega^2\right)^{\frac{1}{2}(m-3)}$;

3: $\mathbf{z}' \leftarrow \left(\omega; \left(\sqrt{1-\omega^2}\right)\mathbf{v}^\top\right)^\top$;

4: $U \leftarrow$ Householder $(\mathbf{e}_1, \mu)$; {Householder transform}

5: **Return**: $U\mathbf{z}'$

---

---

**Algorithm 2** Householder transform

---

**Input**: mean $\mu$, modal vector $\mathbf{e}_1$.

1: $\mathbf{u}' \leftarrow \mathbf{e}_1 - \mu$;

2: $\mathbf{u} \leftarrow \frac{\mathbf{u}'}{\|\mathbf{u}'\|}$;

3: $U \leftarrow \mathbb{I} - 2\mathbf{u}\mathbf{u}^\top$;

4: **Return**: $U$

---

Let $a = e^{-\mathrm{i}\frac{\theta_0+\theta_2}{2}}\cos\frac{\theta_1}{2}$ and $b = e^{\mathrm{i}\frac{\theta_2-\theta_0}{2}}\sin\frac{\theta_1}{2}$, then the unitary matrix $V$ becomes:

$$V = \begin{bmatrix} e^{-\mathrm{i}\frac{\theta_0+\theta_2}{2}}\cos\frac{\theta_1}{2} & -e^{\mathrm{i}\frac{\theta_0-\theta_2}{2}}\sin\frac{\theta_1}{2} \\ e^{\mathrm{i}\frac{\theta_2-\theta_0}{2}}\sin\frac{\theta_1}{2} & e^{\mathrm{i}\frac{\theta_0+\theta_2}{2}}\cos\frac{\theta_1}{2} \end{bmatrix}. \tag{26}$$

Decomposing the matrix $V$, we can obtain:

$$V = \begin{bmatrix} e^{-\mathrm{i}\frac{\theta_2}{2}} & 0 \\ 0 & e^{\mathrm{i}\frac{\theta_2}{2}} \end{bmatrix} \begin{bmatrix} \cos\frac{\theta_1}{2} & -\sin\frac{\theta_1}{2} \\ \sin\frac{\theta_1}{2} & \cos\frac{\theta_1}{2} \end{bmatrix} \begin{bmatrix} e^{-\mathrm{i}\frac{\theta_0}{2}} & 0 \\ 0 & e^{\mathrm{i}\frac{\theta_0}{2}} \end{bmatrix}. \tag{27}$$

Due to

$$\mathbf{R_y}(\theta) = \begin{bmatrix} \cos(\frac{\theta}{2}) & -\sin(\frac{\theta}{2}) \\ \sin(\frac{\theta}{2}) & \cos(\frac{\theta}{2}) \end{bmatrix}, \mathbf{R_z}(\theta) = \begin{bmatrix} e^{-\mathrm{i}\frac{\theta}{2}} & 0 \\ 0 & e^{\mathrm{i}\frac{\theta}{2}} \end{bmatrix}, \tag{28}$$

the matrix $V$ can be decomposed as the sequence of parameterized rotation gates $\mathbf{R_z}\mathbf{R_y}\mathbf{R_z}$, *i.e.*,

$$V = \mathbf{R_z}(\theta_2)\mathbf{R_y}(\theta_1)\mathbf{R_z}(\theta_0). \tag{29}$$

Therefore, an arbitrary unitary matrix can be represented by a sequence of $\mathbf{R_z}$, $\mathbf{R_y}$ and $\mathbf{R_z}$ gates, and a phase $e^{\mathrm{i}\alpha}$. Because it does not affect the outcome of our experiment, we discard the global phase as consensus. Thus, we use the combination of $\mathbf{R_z}$, $\mathbf{R_y}$ and $\mathbf{R_z}$ in the trainable layer. □

## F   Discussions of Latent Space

**Trace out and quantum state tomography** The *tracing out* operation is similar to marginalizing out specific $q_B$ binary dimensions of an $q$-dimensional probability distribution when treating the squared amplitudes of $|\psi\rangle_{AB}$ as a distribution. After tracing out $q_B$ qubits, the dimension of the quantum state comes to $2^{q_A}$.

Readout of the quantum state may require an exponentially large number of measurements. However, there are many methods proposed to tackle this problem. For instance, Schmale et al. [40] propose an efficient quantum state tomography with convolutional neural networks. Moreover, Rambach et al. [42] provide a practical, efficient, and robust self-guided tomography for measuring high dimensional quantum states.

**Rejection Sampling** Here we provide the sampling procedure in Algorithm 1.

## G   Supplementary Experiments

**Metrics.**   Validity is defined as the percentage of molecular graphs that do not violate chemical valency rules. Uniqueness measures the proportion of molecules that have different SMILES

Table 3: Hyperparameter choices of and the training phase settings.

| | Hyperparameters | Values |
|---|---|---|
| Ansatz | # Quantum parametric layer (L) | 8 |
| | # Qubits in subsystem B ($q_B$) | 2 |
| | Concentration parameter in vMF ($\kappa$) | 10 |
| | Generate threshold ($T$) | 0.2 |
| Training | Batch Size | 512 |
| | # Epochs | 50 |
| | Optimizer | Adam |
| | Learning Rate | 0.01 |
| | Weight Decay | 1e-4 |

Table 4: Comparison of MMD distances for bond length distributions in different methods.

| Method | MMD distances ↓ | | | | | | | |
|---|---|---|---|---|---|---|---|---|
| | C-C | C-N | C-O | C-F | C=C | N-N | N-O | C=O | Average |
| MLP-VAE [21] | 2.85 | 7.39 | 2.65 | 4.12 | 1.97 | 7.26 | 2.14 | 1.49 | 3.73 |
| E-NFs [22] | 0.56 | 0.18 | 1.02 | 3.31 | 2.99 | 1.05 | 0.69 | 1.76 | 1.45 |
| G-SchNet [50] | 0.18 | 0.08 | 0.32 | 0.184 | 5.34 | 0.25 | 0.25 | 1.58 | 1.02 |
| G-SphereNet [23] | 1.02 | 0.17 | 0.78 | 0.23 | 2.47 | 1.65 | 0.91 | 0.22 | 0.93 |
| EDM [49] | 0.04 | 0.03 | 0.06 | 0.34 | 5.14 | 0.17 | 0.07 | 1.32 | 0.90 |
| QVAE-Mole | 2.26 | 1.77 | 2.88 | 1.70 | 2.35 | 1.85 | 2.26 | 3.63 | 2.34 |
| QVAE-Mole (fidelity loss) | 2.69 | 2.38 | 1.79 | 3.42 | 3.05 | 3.41 | 3.34 | 4.46 | 3.07 |

strings (which implies that they are non-isomorphic). Novelty measures the proportion of generated molecules that are not in the training set. To evaluate the generated molecular geometry, we evaluate by the Maximum Mean Discrepancy (MMD) [48] distances of bond length distributions. Specifically, for a specific type of bond, we extract its length distribution from both the generated geometries and the dataset's geometries individually. We then calculate the statistical discrepancy between these two length distributions using the MMD distance. Avg.MMD is the average MMD of all bond types.

**Target properties.** SA represents the difficulty of drug synthesis, with higher values indicating easier synthesis as it is normalized between 0 and 1. QED quantifies the likelihood of a molecule being a potential drug candidate. LogP indicates the molecule's partition coefficient between octanol and water, where logP values between -0.4 and 5.6 are considered favorable for drug candidates [79]. HUMO-LUMO gap signifies the energy difference between the highest occupied molecular orbital (HOMO) and the lowest unoccupied molecular orbital (LUMO).

**Training Details.** In the TorchQuantum simulator, it is known that the running of quantum circuits is simulated with unitary operations, and the unitary operations can be implemented through PyTorch's underlying tensor operations. Thus obtaining the quantum state vector in the simulator only requires one forward unitary computation, which makes the entire process differentiable in TorchQuantum. In this way, we can directly get the gradient of quantum parameters, perform backpropagation, and train the model. We use stochastic gradient descent with Adam optimizer [80] to train our model for a maximum of 50 epochs with a batch size of 512 and a learning rate of 0.01. The choice of hyperparameters is shown in Table 3. The parameters in both the training and inference stages of QVAE-Mole and QCVAE-Mole remain the same.

**Evaluate 3-D molecular geometry.** Following [23], we compute the MMD distances on eight most frequently appeared types of chemical bonds, including carbon-carbon single bonds (C-C), carbon-nitrogen single bonds (C-N), carbon-oxygen single bonds (C-O), carbon-carbon double bonds (C=C), nitrogen-nitrogen single bond (N-N), nitrogen-oxygen single bond (N-O), carbon-oxygen double bond (C=O). All metrics are computed from 10k generated molecular geometries and the results are shown in Table 4. Since SQ-VAE and QGAN-HG cannot generate 3-D molecules, here we only compare QCVAE-Mole with classic SOTA methods.

# H   Impact Statements

Quantum computing can play a significant role beyond molecule generation. Thus we shall be cautious about this technology especially when quantum generative AI could have a broad impact on society.

